# Learning Fuzzy Rule-Based Neural Networks for Control

Charles M. Higgins and Rodney M. Goodman
Department of Electrical Engineering, 116-81
California Institute of Technology
Pasadena, CA 91125

## Abstract

A three-step method for function approximation with a fuzzy system is proposed. First, the membership functions and an initial rule representation are learned; second, the rules are compressed as much as possible using information theory; and finally, a computational network is constructed to compute the function value. This system is applied to two control examples: learning the truck and trailer backer-upper control system, and learning a cruise control system for a radio-controlled model car.

## 1   Introduction

Function approximation is the problem of estimating a function from a set of examples of its independent variables and function value. If there is prior knowledge of the type of function being learned, a mathematical model of the function can be constructed and the parameters perturbed until the best match is achieved. However, if there is no prior knowledge of the function, a model-free system such as a neural network or a fuzzy system may be employed to approximate an arbitrary nonlinear function. A neural network's inherent parallel computation is efficient for speed; however, the information learned is expressed only in the weights of the network. The advantage of fuzzy systems over neural networks is that the information learned is expressed in terms of linguistic rules. In this paper, we propose a method for learning a complete fuzzy system to approximate example data. The membership functions and a minimal set of rules are constructed automatically from the example data, and in addition the final system is expressed as a computational

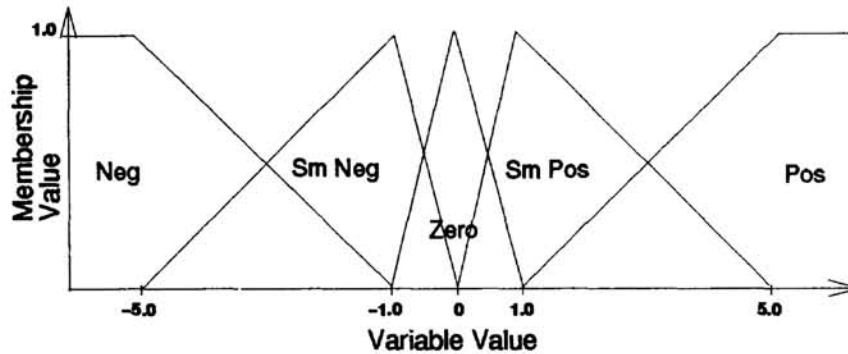

Figure 1: Membership function example

(neural) network for efficient parallel computation of the function value, combining the advantages of neural networks and fuzzy systems. The proposed learning algorithm can be used to construct a fuzzy control system from examples of an existing control system's actions.

Hereafter, we will refer to the function value as the output variable, and the independent variables of the function as the input variables.

## 2   Fuzzy Systems

In a fuzzy system, a function is expressed in terms of membership functions and rules. Each variable has membership functions which partition its range into overlapping classes (see figure 1). Given these membership functions for each variable, a function may be expressed by making rules from the input space to the output space and smoothly varying between them.

In order to simplify the learning of membership functions, we will specify a number of their properties beforehand. First, we will use piecewise linear membership functions. We will also specify that membership functions are *fully overlapping*; that is, at any given value of the variable the total membership sums to one. Given these two properties of the membership functions, we need only specify the positions of the peaks of the membership functions to completely describe them.

We define a fuzzy rule as *if y then X*, where $y$ (the condition side) is a conjunction in which each clause specifies an input variable and one of the membership functions associated with it, and $X$ (the conclusion side) specifies an output variable membership function.

## 3   Learning a Fuzzy System from Example Data

There are three steps in our method for constructing a fuzzy system: first, learn the membership functions and an initial rule representation; second, simplify (compress) the rules as much as possible using information theory; and finally, construct a computational network with the rules and membership functions to calculate the function value given the independent variables.

## 3.1    Learning the Membership Functions

Before learning, two parameters must be specified. First, the maximum allowable RMS error of the approximation from the example data; second, the maximum number of membership functions for each variable. The system will not exceed this number of membership functions, but may use fewer if the error is reduced sufficiently before the maximum number is reached.

### 3.1.1    Learning by Successive Approximation to the Target Function

The following procedure is performed to construct membership functions and a set of rules to approximate the given data set. All of the rules in this step are *cell-based*, that is, they have a condition for every input variable; there is a rule for every combination of input variables (*cell*).

We begin with input membership functions at input extrema. The closest example point to each "corner" of the input space is found and a membership function for the output is added at its value at the corner point. The initial rule set contains a rule for each corner, specifying the closest output membership function to the actual value at that corner.

We now find the example point with the greatest RMS error from the current model and add membership functions *in each variable* at that point. Next, we construct a new set of rules to approximate the function. Constructing rules simply means determining the output membership function to associate with each cell. While constructing this rule set, we also add any output membership functions which are needed. The best rule for a given cell is found by finding the closest example point to the rule (recall each rule specifies a point in the input space). If the output value at this point is "too far" from the closest output membership function value, this output value is added as a new output membership. After this addition has been made, if necessary, the closest output membership function to the value at the closest point is used as the conclusion of the rule. At this point, if the error threshold has been reached or all membership functions are full, we exit. Otherwise, we go back to find the point with the greatest error from the model and iterate again.

## 3.2    Simplifying the Rules

In order to have as simple a fuzzy system as possible, we would like to use the minimum possible number of rules. The initial cell-based rule set can be "compressed" into a minimal set of rules; we propose the use of an information-theoretic algorithm for induction of rules from a discrete data set [1] for this purpose. The key to the use of this method is the interpretation of each of the original rules as a discrete example. The rule set becomes a discrete data set which is input to a rule-learning algorithm. This algorithm learns the best rules to describe the data set.

There are two components of the rule-learning scheme. First, we need a way to tell which of two candidate rules is the best. Second, we need a way to search the space of all possible rules in order to find the best rules without simply checking every rule in the search space.

### 3.2.1    Ranking Rules

Smyth and Goodman[2] have developed an information-theoretic measure of rule value with respect to a given discrete data set. This measure is known as the j-measure; defining a rule as *if y then X*, the j-measure can be expressed as follows:

$$j(X|y) = p(X|y) \log_2(\frac{p(X|y)}{p(X)}) + p(\bar{X}|y) \log_2(\frac{p(\bar{X}|y)}{p(\bar{X})})$$

[2] also suggests a modified rule measure, the J-measure:

$$J(X|y) = p(y)j(X|y)$$

This measure discounts rules which are not as useful in the data set in order to remove the effects of "noise" or randomness. The probabilities in both measures are computed from relative frequencies counted in the given discrete data set.

Using the j-measure, examples will be combined only when no error is caused in the prediction of the data set. The J-measure, on the other hand, will combine examples even if some prediction ability of the data is lost. If we simply use the j-measure to compress our original rule set, we don't get significant compression. However, we can only tolerate a certain margin of error in prediction of our original rule set and maintain the same control performance. In order to obtain compression, we wish to allow some error, but not so much as the J-measure will create. We thus propose the following measure, which allows a gradual variation of the amount of noise tolerance:

$$L(X|y) = f(p(y), \alpha)\, j(X|y) \quad \text{where} \quad f(x, \alpha) = \frac{1 - e^{-\alpha x}}{1 - e^{-\alpha}}$$

The parameter $\alpha$ may be set at $0^+$ to obtain the J-measure since $f(x, 0^+) = x$ or at $\infty$ to obtain the j-measure, since $f(x, \infty) = 1$ $(x > 0)$. Any value of $\alpha$ between 0 and $\infty$ will result in an amount of compression between that of the J-measure and the j-measure; thus if we are able to tolerate some error in the prediction of the original rule set, we can obtain more compression than the j-measure could give us, but not as much as the J-measure would require. We show an example of the variation of $\alpha$ for the truck backer-upper control system in section 4.1.

### 3.2.2    Searching for the Best Rules

In [1], we presented an efficient method for searching the space of all possible rules to find the most representative ones for discrete data sets. The basic idea is that each example is a very specific (and quite perfect) rule. However, this rule is applicable to only one example. We wish to generalize this very specific rule to cover as many examples as possible, while at the same time keeping it as correct as possible. The goodness-measures shown above are just the tool for doing this. If we calculate the "goodness" of all the rules generated by *removing a single input variable* from the very specific rule, then we will be able to tell if any of the slightly more general rules generated from this rule are better. If so, we take the best and continue in this manner until no more general rule with a higher "goodness" exists. When we have performed this procedure on the very specific rule generated from each example (and removed duplicates), we will have a set of rules which represents the data set.

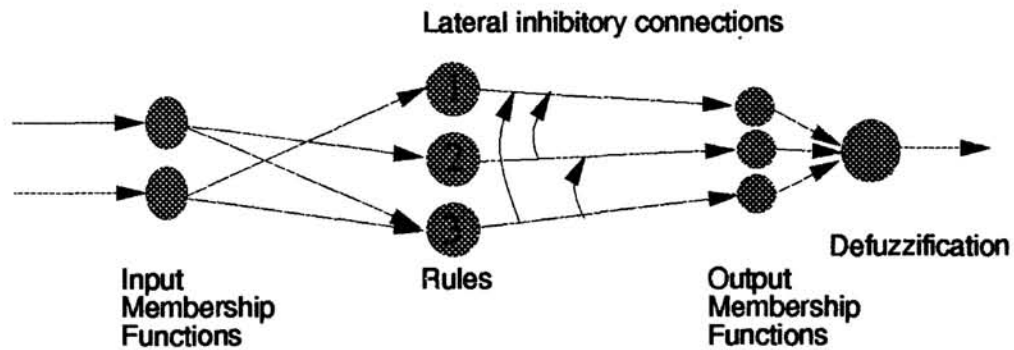

Figure 2: Computational network constructed from fuzzy system

## 3.3    Constructing a Network

Constructing a computational network to represent a given fuzzy system can be accomplished as shown in figure 2. From input to output, layers represent input membership functions, rules, output membership functions, and finally defuzzification. A novel feature of our network is the lateral links shown in figure 2 between the outputs of various rules. These links allow inference with dependent rules.

### 3.3.1    The Layers of the Network

The first layer contains a node for every input membership function used in the rule set. Each of these nodes responds with a value between zero and one to a certain region of the input variable range, implementing a single membership function. The second layer contains a node for each rule – each of these nodes represents a fuzzy AND, implemented as a product. The third layer contains a node for every output membership function. Each of these nodes sums the outputs from each rule that concludes that output fuzzy set. The final node simply takes the output memberships collected in the previous layer and performs a defuzzification to produce the final crisp output by normalizing the weights from each output node and performing a convex combination with the peaks of the output membership functions.

### 3.3.2    The Problem with Dependent Rules and a Solution

There is a problem with the standard fuzzy inference techniques when used with dependent rules. Consider a rule whose conditions are all contained in a more specific rule (i.e. one with more conditions) which contradicts its conclusion. Using standard fuzzy techniques, the more general rule will drive the output to an intermediate value between the two conclusions. What we really want is that a more general rule dependent on a more specific rule should only be allowed to fire *to the degree that the more specific rule is not firing*. Thus the degree of firing of the more specific rule should gate the maximum firing allowed for the more general rule. This is expressed in network form in the links between the rule layer and the output membership functions layer. The lateral arrows are inhibitory connections which take the value at their input, invert it (subtract it from one), and multiply it by the value at their output.

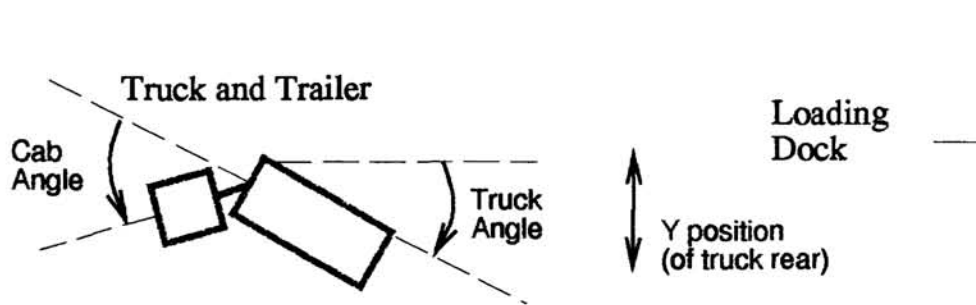

Figure 3: The truck and trailer backer-upper problem

# 4   Experimental Results

In this section, we show the results of two experiments: first, a truck backer-upper in simulation; and second, a simple cruise controller for a radio-controlled model car constructed in our laboratory.

## 4.1   Truck and Trailer Backer-Upper

Jenkins and Yuhas [3] have developed by hand a very efficient neural network for solving the problem of backing up a truck and trailer to a loading dock. The truck and trailer backer-upper problem is parameterized in figure 3.

The function approximator system was trained on 225 example runs of the Yuhas controller, with initial positions distributed symmetrically about the field in which the truck operates. In order to show the effect of varying the number of membership functions, we have fixed the maximum number of membership functions for the y position and cab angle at 5 and set the maximum allowable error to zero, thus guaranteeing that the system will fill out all of the allowed membership functions. We varied the maximum number of truck angle membership functions from 3 to 9. The effects of this are shown in figure 4. Note that the error decreases sharply and then holds constant, reaching its minimum at 5 membership functions. The Yuhas network performance is shown as a horizontal line. At its best, the fuzzy system performs slightly better than the system it is approximating.

For this experiment, we set a goal of 33% rule compression. We varied the parameter $\alpha$ in the $L$-measure for each rule set to get the desired compression. Note in figure 4 the performance of the system with compressed rules. The performance is in every case almost identical to that of the original rule sets. The number of rules and the amount of rule compression obtained can be seen in table 1.

## 4.2   Cruise Controller

In this section, we describe the learning of a cruise controller to keep a radio controlled model car driving at a constant speed in a circle. We designed a simple PD controller to perform this task, and then learned a fuzzy system to perform the same task. This example is not intended to suggest that a fuzzy system should replace a simple PD controller, since the fuzzy system may represent far more complex

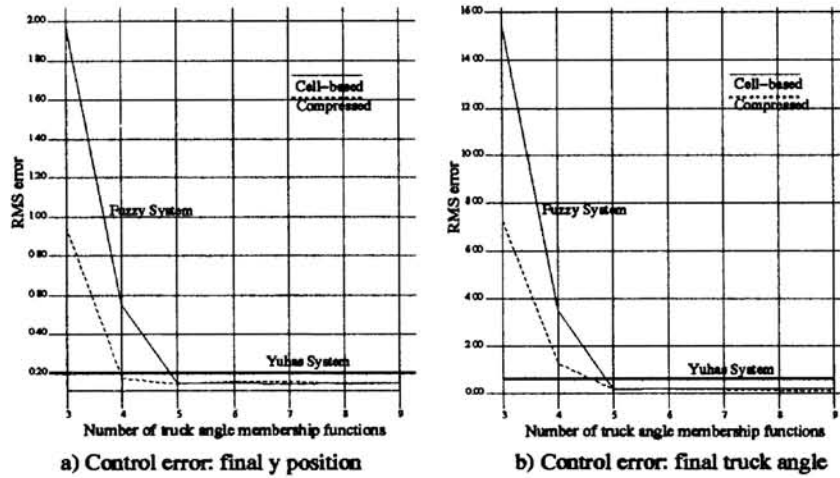

a) Control error: final y position    b) Control error: final truck angle

Figure 4: Results of experiments with the truck backer-upper

| | | Number of truck angle membership functions | | | | | | |
|---|---|---|---|---|---|---|---|---|
| | | 3 | 4 | 5 | 6 | 7 | 8 | 9 |
| Number of Rules | Cell-Based | 75 | 100 | 125 | 150 | 175 | 200 | 225 |
| | Compressed | 48 | 67 | 86 | 100 | 114 | 138 | 154 |
| Compression | | 36% | 33% | 31% | 33% | 35% | 31% | 32% |

Table 1: Number of rules and compression figures for learned TBU systems

functions, but rather to show that the fuzzy system can learn from real control data and operate in real-time.

The fuzzy system was trained on 6 runs of the PD controller which included runs going forward and backward, and conditions in which the car's speed was perturbed momentarily by blocking the car or pushing it. Figure 5 shows the error trajectory of both the hand-crafted PD and learned fuzzy control systems from rest. The car builds speed until it reaches the desired set point with a well-damped response, then holds speed for a while. At a later time, an obstacle was placed in the path of the car to stop it and then removed; figure 5 shows the similar recovery responses of both systems. It can be seen from the numerical results in table 2 that the fuzzy system performs as well as the original PD controller.

No compression was attempted because the rule sets are already very small.

| | PD Controller | Learned Fuzzy System |
|---|---|---|
| Time from 90% error to 10% error (s) | 0.9 | 0.7 |
| RMS error at steady state (uncal) | 59 | 45 |
| Time to correct after obstacle (s) | 6.2 | 6.2 |

Table 2: Analysis of cruise control performance

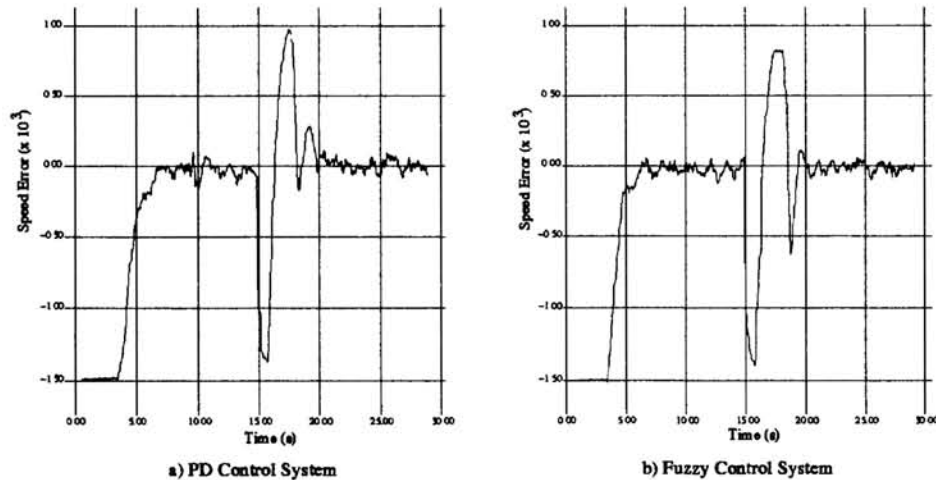

Figure 5: Performance of PD controller vs. learned fuzzy system

## 5    Summary and Conclusions

We have presented a method which, given examples of a function and its independent variables, can construct a computational network based on fuzzy logic to predict the function given the independent variables. The user must only specify the maximum number of membership functions for each variable and the maximum RMS error from the example data.

The final fuzzy system's actions can be explicitly explained in terms of rule firings. If a system designer does not like some aspect of the learned system's performance, he can simply change the rule set and the membership functions to his liking. This is in direct contrast to a neural network system, in which he would have no recourse but another round of training.

### Acknowledgements

This work was supported in part by Pacific Bell, and in part by DARPA and ONR under grant no. N00014-92-J-1860.

### References

[1] C. Higgins and R. Goodman, "Incremental Learning using Rule-Based Neural Networks," *Proceedings of the International Joint Conference on Neural Networks*, vol. 1, 875-880, July 1991.

[2] R. Goodman, C. Higgins, J. Miller, P. Smyth, "Rule-Based Networks for Classification and Probability Estimation," *Neural Computation* 4(6), 781-804, November 1992.

[3] R. Jenkins and B. Yuhas, "A Simplified Neural-Network Solution through Problem Decomposition: The Case of the Truck Backer-Upper," *Neural Computation* 4(5), 647-9, September 1992.
